# A nonparametric variable clustering model

**Konstantina Palla**[*]
University of Cambridge
kp376@cam.ac.uk

**David A. Knowles**[*]
Stanford University
davidknowles@cs.stanford.edu

**Zoubin Ghahramani**
University of Cambridge
zoubin@eng.cam.ac.uk

## Abstract

Factor analysis models effectively summarise the covariance structure of high dimensional data, but the solutions are typically hard to interpret. This motivates attempting to find a disjoint partition, i.e. a simple clustering, of observed variables into highly correlated subsets. We introduce a Bayesian non-parametric approach to this problem, and demonstrate advantages over heuristic methods proposed to date. Our Dirichlet process variable clustering (DPVC) model can discover block-diagonal covariance structures in data. We evaluate our method on both synthetic and gene expression analysis problems.

## 1   Introduction

Latent variables models such as principal components analysis (Pearson, 1901; Hotelling, 1933; Tipping and Bishop, 1999; Roweis, 1998) and factor analysis (Young, 1941) are popular for summarising high dimensional data, and can be seen as modelling the covariance of the observed dimensions. Such models may be used for tasks such as collaborative filtering, dimensionality reduction, or data exploration. For all these applications sparse factor analysis models can have advantages in terms of both predictive performance and interpretability (Fokoue, 2004; Fevotte and Godsill, 2006; Carvalho et al., 2008). For example, data exploration might involve investigating which variables have significant loadings on a shared factor, which is aided if the model itself is sparse. However, even using sparse models interpreting the results of a factor analysis can be non-trivial since a variable will typically have significant loadings on multiple factors.

As a result of these problems researchers will often simply cluster variables using a traditional agglomerative hierarchical clustering algorithm (Vigneau and Qannari, 2003; Duda et al., 2001). Interest in variable clustering exists in many applied fields, e.g. chemistry (Basak et al., 2000a,b) and acturial science (Sanche and Lonergan, 2006). However, it is most commonly applied to gene expression analysis (Eisen et al., 1998; Alon et al., 1999; D'haeseleer et al., 2005), which will also be the focus of our investigation. Note that variable clustering represents the opposite regime to the usual clustering setting where we partition samples rather than dimensions (but of course a clustering algorithm can be made to work like this simply by transposing the data matrix). Typical clustering algorithms, and their probabilistic mixture model analogues, consider how similar entities are (e.g. in terms of Euclidean distance) rather how correlated they are, which would be closer in spirit to the ability of factor analysis to model covariance structure. While using correlation distance (one minus the Pearson correlation coefficient) between variables has been proposed for clustering genes with heuristic methods, the corresponding probabilistic model appears not to have been explored to the best of our knowledge.

---

[*]These authors contributed equally to this work

To address the general problem of variable clustering we develop a simple Bayesian nonparametric model which partitions observed variables into sets of highly correlated variables. We denote our method DPVC for "Dirichlet Process Variable Clustering". DPVC exhibits the usual advantages over heuristic methods of being both probabilistic and non-parametric: we can naturally handle missing data, learn the appropriate number of clusters from data, and avoid overfitting.

The paper is organised as follows. Section 2 describes the generative process. In Section 3 we note relationships to existing nonparametric sparse factor analysis models, Dirichlet process mixture models, structure learning with hidden variables, and the closely related "CrossCat" model (Shafto et al., 2006). In Section 4 we describe efficient MCMC and variational Bayes algorithms for performing posterior inference in DPVC, and point out computational savings resulting from the simple nature of the model. In Section 5 we present results on synthetic data where we test the method's ability to recover a "true" partitioning, and then focus on clustering genes based on gene expression data, where we assess predictive performance on held out data. Concluding remarks are given in Section 6.

## 2  The Dirichlet Process Variable Clustering Model

Consider observed data $\{\mathbf{y}_n \in \mathbb{R}^D : n = 1, .., N\}$ where we have $D$ observed dimensions and $N$ samples. The $D$ observed dimensions correspond to measured variables for each sample, and our goal is to cluster these variables. We partition the observed dimensions $d = \{1, ..., D\}$ according to the Chinese restaurant process (Pitman, 2002, CRP). The CRP defines a distribution over partitionings (clustering) where the maximum possible number of clusters does not need to be specified a priori. The CRP can be described using a sequential generative process: $D$ customers enter a Chinese restaurant one at a time. The first customer sits at some table and each subsequent customer sits at table $k$ with $m_k$ current customers with probability proportional to $m_k$, or at a new table with probability proportional to $\alpha$, where $\alpha$ is a parameter of the CRP. The seating arrangement of the customers at tables corresponds to a partitioning of the $D$ customers. We write

$$(c_1, ..., c_D) \sim \text{CRP}(\alpha), \qquad c_d \in \mathbb{N} \tag{1}$$

where $c_d = k$ denotes that variable $d$ belongs to cluster $k$. The CRP partitioning allows each dimension to belong only to one cluster. For each cluster $k$ we have a single latent factor

$$x_{kn} \sim N(0, \sigma_x^2) \tag{2}$$

which models correlations between the variables in cluster $k$. Given these latent factors, real valued observed data can be modeled as

$$y_{dn} = g_d x_{c_d n} + \epsilon_{dn} \tag{3}$$

where $g_d$ is a factor loading for dimension $d$, and $\epsilon_{dn} \sim N(0, \sigma_d^2)$ is Gaussian noise. We place a Gaussian prior $N(0, \sigma_g^2)$ on every element $g_d$ independently. It is straightforward to generalise the model by substituting other noise models for Equation 3, for example using a logistic link for binary data $y_{dn} \in \{0, 1\}$. However, in the following we will focus on the Gaussian case.

To improve the flexibility of the model, we put Inverse Gamma priors on $\sigma_g^2$ and $\sigma_d^2$ and a Gamma prior on the CRP concentration parameter $\alpha$ as follows:

$$\alpha \sim \mathcal{G}(1, 1)$$
$$\sigma_g^2 \sim \mathcal{IG}(1, 1)$$
$$\sigma_d^2 \sim \mathcal{IG}(1, 0.1)$$

Note that we fix $\sigma_x = 1$ due to the scale ambiguity in the model.

## 3  Related work

Since DPVC is a hybrid mixture/factor analysis model there is of course a wealth of related work, but we aim to highlight a few interesting connections here.

DPVC can be seen as a simplification of the infinite factor analysis models proposed by Knowles and Ghahramani (2007) and Rai and Daumé III (2008), which we will refer to as Non-parametric

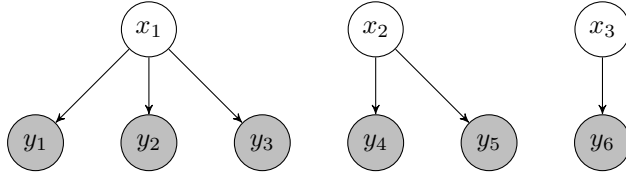

Figure 1: Graphical model structure that could be learnt using the model, corresponding to cluster assignments $\mathbf{c} = \{1, 1, 1, 2, 2, 3\}$. Gray nodes represent the $D = 6$ observed variables $y_d$ and white nodes represent the $K = 3$ latent variables $x_k$.

Sparse Factor Analysis (NSFA). Where they used the Indian buffet process to allow dimensions to have non-zero loadings on multiple factors, we use the Chinese restaurant process to explicitly enforce that a dimension can be explained by only one factor. Obviously this will not be appropriate in all circumstances, but where it is appropriate we feel it allows easier interpretation of the results. To see the relationship more clearly, introduce the indicator variable $z_{dk} = \mathbb{I}[c_d = k]$. We can then write our model as

$$\mathbf{y}_n = (\mathbf{G} \cdot \mathbf{Z})\mathbf{x}_n + \epsilon_n \tag{4}$$

where $\mathbf{G}$ is a $D \times K$ Gaussian matrix, and $\cdot$ denotes elementwise multiplication. Replacing our Chinese restaurant process prior on $\mathbf{Z}$ with an Indian buffet prior recovers an infinite factor analysis model. Equation 4 has the form of a factor analysis model. It is straightforward to show that the conditional covariance of $\mathbf{y}$ given the factor loading matrix $\mathbf{W} := \mathbf{G} \cdot \mathbf{Z}$ is $\sigma_x^2 \mathbf{W}\mathbf{W}^T + \sigma_\epsilon^2 \mathbf{I}$. Analogously for DPVC we find

$$\mathrm{cov}(y_{dn}, y_{d'n} | \mathbf{G}, \mathbf{c}) = \left\{ \begin{array}{ll} \sigma_x^2 g_d g_{d'} + \sigma_d^2 \delta_{dd'}, & c_d = c_{d'} \\ 0, & \text{otherwise} \end{array} \right. \tag{5}$$

Thus we see the covariance structure implied by DPVC is block diagonal: only dimensions belonging to the same cluster have non-zero covariance.

The obvious probabilistic approach to clustering genes would be to simply apply a Dirichlet process mixture (DPM) of Gaussians, but considering the genes (our dimensions) as samples, and our samples as "features" so that the partitioning would be over the genes. However, this approach would not achieve the desired result of clustering *correlated* variables, and would rather cluster together variables close in terms of Euclidean distance. For example two variables which have the relationship $y_d = a y_{d'}$ for $a = -1$ (or $a = 2$) are perfectly correlated but not close in Euclidean space; a DPM approach would likely fail to cluster these together. Also, practitioners typically choose either to use restrictive diagonal Gaussians, or full covariance Gaussians which result in considerably greater computational cost than our method (see Section 4.3).

DPVC can also be seen as performing a simple form of structure learning, where the observed variables are partitioned into groups explained by a single latent variable. This is subset of the structures considered in Silva et al. (2006), but we maintain uncertainty over the structure using a fully Bayesian analysis. Figure 1 illustrates this idea.

DPVC is also closely related to CrossCat (Shafto et al., 2006). CrossCat also uses a CRP to partition variables into clusters, but then uses a second level of independent CRPs to model the dependence of variables within a cluster. In other words whereas the latent variables $\mathbf{x}$ in Figure 1 are discrete variables (indicating cluster assignment) in CrossCat, they are continuous variables in DPVC corresponding to the latent factors. For certain data the CrossCat model may be more appropriate but our simple factor analysis model is more computationally tractable and often has good predictive performance as well. The model of Niu et al. (2012) is related to CrossCat in the same way that NSFA is related to DPVC, by allowing an observed dimension to belong to multiple features using the IBP rather than only one cluster using the CRP.

## 4   Inference

We demonstrate both MCMC and variational inference for the model.

| **Algorithm 1** Marginal conditional | **Algorithm 2** Successive conditional |
|---|---|
| 1: **for** $m = 1$ to $M$ **do** | 1: $\theta^{(1)} \sim P(\theta)$ |
| 2: $\quad \theta^{(m)} \sim P(\theta)$ | 2: $Y^{(1)} \sim P(Y\|\theta^{(1)})$ |
| 3: $\quad Y^{(m)} \sim P(Y\|\theta^{(m)})$ | 3: **for** $m = 2$ to $M$ **do** |
| 4: **end for** | 4: $\quad \theta^{(m)} \sim Q(\theta\|\theta^{(m-1)}, Y^{(m-1)})$ |
| | 5: $\quad Y^{(m)} \sim P(Y\|\theta^{(m)})$ |
| | 6: **end for** |

## 4.1 MCMC

We use a partially collapsed Gibbs sampler to explore the posterior distribution over all latent variables $\mathbf{g}, \mathbf{c}, \mathbf{X}$ as well as hyperparameters $\sigma_d^2, \sigma_g^2$ and $\alpha$. The Gibbs update equations for the factor loadings $\mathbf{g}$, factors $\mathbf{X}$, noise variance $\sigma_d^2$ and $\sigma_g^2$ are standard, and therefore only sketched out below with the details deferred to supplementary material. The Dirichlet concentration parameter $\alpha$ is sampled using slice sampling (Neal, 2003). We sample the cluster assignments $\mathbf{c}$ using Algorithm 8 of Neal (2000), with $\mathbf{g}$ integrated out but instantiating $\mathbf{X}$. Updating the factor loading matrix $\mathbf{G}$ is done elementwise, sampling from

$$g_{dk}|\mathbf{Y}, \mathbf{G}_{-dk}, \mathbf{C}, \mathbf{X}, \sigma_g, \sigma_x, \sigma_d, \alpha \sim \mathcal{N}(\mu_g^*, \lambda_g^{-1}) \tag{6}$$

The factors $\mathbf{X}$ can be jointly sampled as

$$\mathbf{X}_{:n}|\mathbf{Y}, \mathbf{G}, \mathbf{C}, \sigma_g, \sigma_x, \sigma_d, \alpha \sim \mathcal{N}(\boldsymbol{\mu}_{\mathbf{X}_{:n}}, \boldsymbol{\Lambda}_{\mathbf{X}_{:n}}^{-1}) \tag{7}$$

When sampling the cluster assignments, $\mathbf{c}$ we found it beneficial to integrate out $\mathbf{g}$, while instantiating $\mathbf{X}$. We require

$$P(c_d = k|y_{d:}, x_{k:}, \sigma_g, \mathbf{c}_{-d}) = P(c_d|\mathbf{c}_{-d}) \int P(y_{d:}|x_{k:}, g_d)p(g_d|\sigma_g)dg_d$$

the calculation of which is given in the supplementary material, along with expressions for $\mu_g^*, \lambda_g, \boldsymbol{\mu}_{\mathbf{X}_{:n}}$ and $\boldsymbol{\Lambda}_{\mathbf{X}_{:n}}$.

We confirm the correctness of our algorithm using the joint distribution testing methodology of Geweke (2004). There are two ways to sample from the joint distribution, $P(Y, \theta)$ over parameters, $\theta = \{\mathbf{g}, \mathbf{c}, \mathbf{X}\}$ and data, $Y$ defined by a probabilistic model such as DPVC. The first we will refer to as "marginal-conditional" sampling, shown in Algorithm 1. Both steps here are straightforward: sampling from the prior followed by sampling from the likelihood model. The second way, referred to as "successive-conditional" sampling is shown in Algorithm 2, where $Q$ represents a single (or multiple) iteration(s) of our MCMC sampler. To validate our sampler we can then check, either informally or using hypothesis tests, whether the samples drawn from the joint $P(Y, \theta)$ in these two different ways appear to have come from the same distribution.

We apply this method to our DPVC sampler with just $N = D = 2$, and all hyperparameters fixed as follows: $\alpha = 1, \sigma_d = 0.1, \sigma_g = 1, \sigma_x = 1$. We draw $10^4$ samples using both the marginal-conditional and successive-conditional procedures. We look at various characteristics of the samples, including the number of clusters and the mean of $\mathbf{X}$. The distribution of the number of features under the successive-conditional sampler matches that under the marginal-conditional sampler almost perfectly. Under the correct successive-conditional sampler the average number of clusters is 1.51 (it should be 1.5): a hypothesis test did not reject the null hypothesis that the means of the two distributions are equal. While this cannot completely guarantee correctness of the algorithm and code, $10^4$ samples is a large number for such a small model and thus gives strong evidence that our algorithm is correct.

## 4.2 Variational inference

We use Variational Message Passing (Winn and Bishop, 2006) under the Infer.NET framework (Minka et al., 2010) to fit an approximate posterior $q$ to the true posterior $p$, by minimising the Kullback-Leibler divergence

$$KL(q||p) = -H[q(\mathbf{v})] - \int q(\mathbf{v}) \log p(\mathbf{v})d\mathbf{v} \tag{8}$$

where $H[q(\mathbf{v})] = -\int q(\mathbf{v})\log q(\mathbf{v})d\mathbf{v}$ is the entropy and $\mathbf{v} = \{w, \mathbf{g}, \mathbf{c}, \mathbf{X}, \sigma_d^2, \sigma_g^2\}$, where $w$ is introduced so that the Dirichlet process can be approximated as

$$w \sim \text{Dirichlet}(\alpha/T, ..., \alpha/T) \tag{9}$$
$$c_d \sim \text{Discrete}(w) \tag{10}$$

where we have truncated to allow a maximum of $T$ clusters. Where not otherwise specified we choose $T = D$ so that every dimension could use its own cluster if this is supported by the data. Note that the Dirichlet process is recovered in the limit $T \to \infty$.

We use a variational posterior of the form

$$q(\mathbf{v}) = q_w(w)q_{\sigma_g^2}(\sigma_g^2) \prod_{d=1}^{D} q_{c_d}(c_d)q_{\sigma_d^2}(\sigma_d^2)q_{g_d|c_d}(g_d|c_d) \prod_{n=1}^{N} q_{x_{nd}}(x_{nd}) \tag{11}$$

where $q_w$ is a Dirichlet distribution, each $q_{c_d}$ is a discrete distribution on $\{1,..,T\}$, $q_{\sigma_g^2}$ and $q_{\sigma_d^2}$ are Inverse Gamma distributions and $q_{nd}$ and $q_{g_d|c_d}$ are univariate Gaussian distributions. We found that using the structured approximation $q_{g_d|c_d}(g_d|c_d)$ where the variational distribution on $g_d$ is conditional on the cluster assignment $c_d$ gave considerably improved performance. Using the representation of the Dirichlet process in Equation 10 this model is conditionally conjugate (i.e. all variables have exponential family distributions conditioned on their Markov blanket) so the VB updates are standard and therefore omitted here.

Due to the symmetry of the model under permutation of the clusters, we are require to somehow break symmetry initially. We experimented with initialising either the variational distribution over the factors $q_{x_{nd}}(x_{nd})$ with mean $N(0, 0.1)$ and variance 1 or each cluster assignments distribution $q_{c_d}(c_d)$ to a sample from a uniform Dirichlet. We found initialising the cluster assignments gave considerably better solutions on average. We also typically ran the algorithm $L = 10$ times and took the solution with the best lower bound on the marginal likelihood.

We also experimented with using Expectation Propagation (Minka, 2001) for this model but found that the algorithm often diverged, presumably because of the multimodality in the posterior. It might be possible to alleviate this using damping, but we leave this to future work.

### 4.3 Computational complexity

DPVC enjoys some computational savings compared to NSFA. For both models sampling the factor loadings matrix is $O(DKN)$, where $K$ is the number of active features/clusters. However, for DPVC sampling the factors $\mathbf{X}$ is considerably cheaper. Calculating the diagonal precision matrix is $O(KD)$ (compared to $O(K^2D)$ for the precision in NSFA), and finding the square root of the diagonal elements is negligible at $O(K)$ (compared to a $O(K^3)$ Cholesky decomposition for NSFA). Finally both models require an $O(DKN)$ operation to calculate the conditional mean of $\mathbf{X}$. Thus where NSFA is $O(DKN + DK^2 + K^3)$, DPVC is only $O(DKN)$, which is the same complexity as $k$-means or Expectation Maximisation (EM) for mixture models with diagonal Gaussian clusters. Note that mixture models with full covariance clusters would typically cost $O(DKN^3)$ in this setting due to the need to perform Cholesky decompositions on $N \times N$ matrices.

## 5 Results

We present results on synthetic data and two gene expression data sets. We show comparisons to $k$-means and hierarchical clustering, for which we use the algorithms provided in the Matlab statistics toolbox. We also compare to our implementation of Bayesian factor analysis (see for example Kaufman and Press (1973) or Rowe and Press (1998)) and the non-parametric sparse factor analysis (NSFA) model of (Knowles and Ghahramani, 2011). We experimented with three publicly available implementations of DPM of Gaussian using full covariance matrices, but found that none of them were sufficiently numerically robust to cope with the high dimensional and sometimes ill conditioned gene expression data analysed in Section 5. To provide a similar comparison we implemented a DPM of diagonal covariance Gaussians using a collapsed Gibbs sampler.

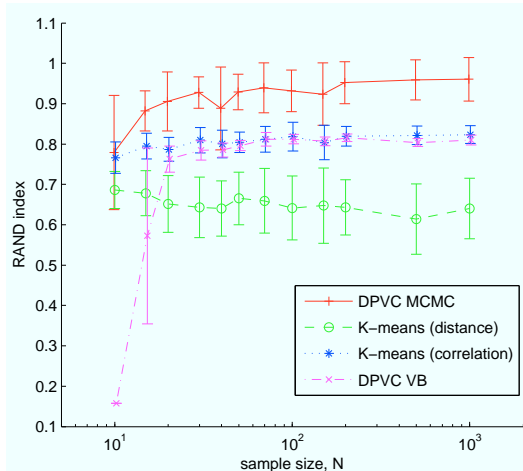

Figure 2: Performance of DPVC compared to $k$-means at recoverying the true partitioning used to simulate the data.

| Dataset | DPVC | NSFA | DPM | FA ($K = 5$) | FA ($K = 10$) | FA ($K = 20$) |
|---|---|---|---|---|---|---|
| Breast cancer | $-0.876 \pm 0.024$ | $-0.634 \pm 0.038$ | $-1.348 \pm 0.108$ | $-1.129 \pm 0.043$ | $-1.275 \pm 0.056$ | $-1.605 \pm 0.072$ |
| Yeast | $-0.849 \pm 0.012$ | $-0.653 \pm 0.061$ | $-1.397 \pm 0.419$ | $-1.974 \pm 1.925$ | $-1.344 \pm 0.165$ | $-1.115 \pm 0.052$ |

Table 1: Predictive performance (mean log predictive loglikelihood over the test elements) results on two gene expression datasets.

## 5.1 Synthetic data

In order to test the ability of the models to recover a true underlying partitioning of the variables into correlated groups we use synthetic data. We generate synthetic data with $D = 20$ dimensions partitioned into $K = 5$ equally sized clusters (of four variables). Within each cluster we sample analoguously to our model: sample $x_{kn} \sim N(0, 1)$ for all $k, n$, then $g_d \sim N(0, 1)$ for all $d$ and finally sample $y_{dn} \sim N(g_d x_{c_d n}, 0.1)$ for all $d, n$. We vary the sample size $N$ and perform 10 repeats for each sample size. We compare $k$-means (with the true number of clusters 5) using Euclidean distance and correlation distance, and DPVC with inference using MCMC or variational Bayes. To compare the inferred and true partitions we calculate the well known Rand index, which varies between 0 and 1, with 1 denoting perfect recovering of the true clustering. The results are shown in Figure 2. We see that the MCMC implementation of DPVC consistently outperforms the $k$-means methods. As expected given the nature of the data simulation, $k$-means using the correlation distance performs better than using Euclidean distance. DPVC VB's performance is somewhat disappointing, suggesting that even the structured variational posterior we use is a poor approximation of the true posterior. We emphasise that $k$-means is given a significant advantage: it is provided with the true number of clusters. In this light, the performance of DPVC MCMC is impressive, and the seemingly poor performance of DPVC VB is more forgivable (DPVC VB used a truncation level $T = D = 20$).

## 5.2 Breast cancer dataset

We assess these algorithms in terms of *predictive performance* on the breast cancer dataset of West et al. (2007), including 226 genes across 251 individuals. The samplers were found to have converged after around 500 samples according to standard multiple chain convergence measures, so 1000 MCMC iterations were used for all models. The predictive log likelihood was calculated using every 10th sample form the final 500 samples. We ran 10 repeats holding out a different random 10% of the the elements of the matrix as test data each time. The results are shown in Table 1. We see that NSFA performs the best, followed by DPVC. This is not surprising and is the price DPVC pays for a more interpretable solution. However, DPVC does outperform both the DPM and the finite (non-sparse) factor analysis models. We also ran DPVC VB on this dataset but its performance was

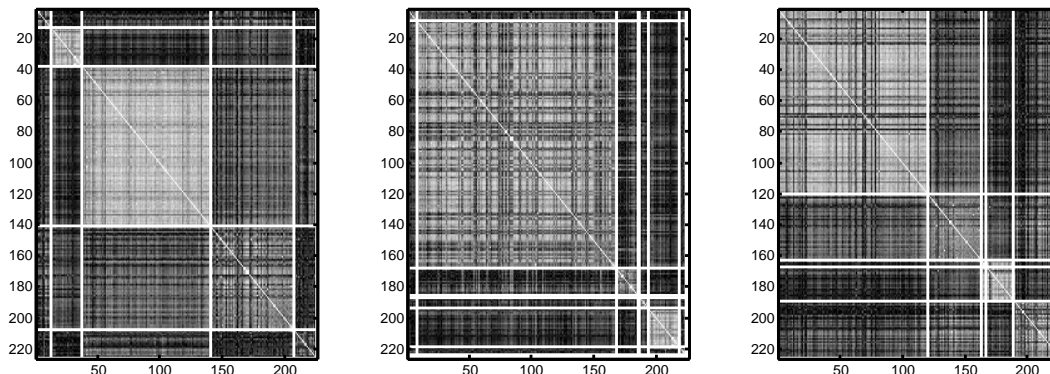

Figure 3: Clustering of the covariance structure. *Left:* $k$-means using correlation distance. *Middle:* Agglomerative heirarchical clustering using average linkage and correlation distance. *Right:* DPVC MCMC.

significantly below that of the MCMC method, with a predictive log likelihood of $-1.154 \pm 0.010$. Performing a Gene Ontology enrichment analysis we find clusters enriched for genes involved in both cell cycle regulation and cell division, which is biologically reasonable in a cancer orientated dataset

On this relatively small dataset it is possible to visualise the $D \times D$ empirical correlation matrix of the data, and investigate what structure our clustering has uncovered, as shown in Figure 3. The genes have been reordered in each plot according three different clusterings coming from $k$-means, hierarchical clustering and DPVC (MCMC, note we show the clustering corresponding to the posterior sample with the highest joint probability). For both $k$-means and hierarchical clustering it was necessary to "tweak" the number of clusters to give a sensible result. Hierarchical clustering in particular appeared to have a strong bias towards putting the majority of the genes in one large cluster/clade. Note that such a visualisation is straightforward only because we have used a clustering based method rather than a factor analysis model, emphasising how partitionings can be more useful summaries of data for certain tasks than low dimensional embeddings.

### 5.3 Yeast in varying environmental conditions

We use the data set of (Gasch et al., 2000), a collection of $N = 175$ non-cell-cycle experiments on *S. cerevisiae* (yeast), including conditions such as heat shock, nitrogen depletion and amino acid starvation. Measurements are available for $D = 6152$ genes. Again we ran 10 repeats holding out a different random $10\%$ of the the elements of the matrix as test data each time. The results shown in Table 1 are broadly consistent with our findings for the breat cancer dataset: DPVC sits between NSFA and the less performant DPM and FA models. Running 1000 iterations of DPVC MCMC on this dataset takes around 1.2 hours on a standard dual core desktop running at 2.5GHz with 4Gb RAM. Unfortunately we were unable to run the VB algorithm on a dataset of this size due to memory constraints.

## 6 Discussion

We have introduced DPVC, a model for clustering variables into highly correlated subsets. While, as expected, we found the predictive performance of DPVC is somewhat worse than that of state of the art nonparametric sparse factor analysis models (e.g. NSFA), DPVC outperforms both nonparametric mixture models and Bayesian factor analysis models when applied to high dimensional data such as gene expression microarrays. For a practitioner we see interpretability as the key advantage of DPVC relative to a model such as NSFA: one can immediately see which groups of variables are correlated, and use this knowledge to guide further analysis. An example use one could envisage would be using DPVC in an analoguous fashion to principal components regression: regressing a dependent variable against the inferred factors $\mathbf{X}$. Regression coefficients would then correspond to the predictive ability of the clusters of variables.

# 7 Acknowledgements

This work was supported by the Engineering and Physical Sciences Research Council (EPSRC) under Grant Number EP/I036575/1 and EP/H019472/1.

# References

Alon, U., Barkai, N., Notterman, D., Gish, K., Ybarra, S., Mack, D., and Levine, A. (1999). Broad patterns of gene expression revealed by clustering analysis of tumor and normal colon tissues probed by oligonucleotide arrays. *Proceedings of the National Academy of Sciences*, 96(12):6745.

Basak, S., Balaban, A., Grunwald, G., and Gute, B. (2000a). Topological indices: their nature and mutual relatedness. *Journal of chemical information and computer sciences*, 40(4):891–898.

Basak, S., Grunwald, G., Gute, B., Balasubramanian, K., and Opitz, D. (2000b). Use of statistical and neural net approaches in predicting toxicity of chemicals. *Journal of Chemical Information and Computer Sciences*, 40(4):885–890.

Carvalho, C. M., Chang, J., Lucas, J. E., Nevins, J. R., Wang, Q., and West, M. (2008). High-dimensional sparse factor modeling: Applications in gene expression genomics. *Journal of the American Statistical Association*, 103(484):1438–1456.

D'haeseleer, P. et al. (2005). How does gene expression clustering work? *Nature biotechnology*, 23(12):1499–1502.

Duda, R. O., Hart, P. E., and Stork, D. G. (2001). *Pattern Classification*. Wiley-Interscience, 2nd edition.

Eisen, M., Spellman, P., Brown, P., Botstein, D., Sherlock, G., Zhang, M., Iyer, V., Anders, K., Botstein, D., Futcher, B., et al. (1998). Gene expression: Clustering. *Proc Natl Acad Sci US A*, 95(25):14863–8.

Fevotte, C. and Godsill, S. J. (2006). A Bayesian approach for blind separation of sparse sources. *Audio, Speech, and Language Processing, IEEE Transactions on*, 14(6):2174–2188.

Fokoue, E. (2004). Stochastic determination of the intrinsic structure in Bayesian factor analysis. Technical report, Statistical and Applied Mathematical Sciences Institute.

Gasch, A., Spellman, P., Kao, C., Carmel-Harel, O., Eisen, M., Storz, G., Botstein, D., and Brown, P. (2000). Genomic expression programs in the response of yeast cells to environmental changes. *Science's STKE*, 11(12):4241.

Geweke, J. (2004). Getting it right. *Journal of the American Statistical Association*, 99(467):799–804.

Hotelling, H. (1933). Analysis of a complex of statistical variables into principal components. *Journal of Educational Psychology*, 24:417–441.

Kaufman, G. M. and Press, S. J. (1973). Bayesian factor analysis. Technical Report 662-73, Sloan School of Management, University of Chicago.

Knowles, D. A. and Ghahramani, Z. (2007). Infinite sparse factor analysis and infinite independent components analysis. In *7th International Conference on Independent Component Analysis and Signal Separation*, volume 4666, pages 381–388. Springer.

Knowles, D. A. and Ghahramani, Z. (2011). Nonparametric Bayesian sparse factor models with application to gene expression modeling. *The Annals of Applied Statistics*, 5(2B):1534–1552.

Minka, T. P. (2001). Expectation propagation for approximate Bayesian inference. In *Conference on Uncertainty in Artificial Intelligence (UAI)*, volume 17.

Minka, T. P., Winn, J. M., Guiver, J. P., and Knowles, D. A. (2010). Infer.NET 2.4.

Neal, R. M. (2000). Markov chain sampling methods for Dirichlet process mixture models. *Journal of computational and graphical statistics*, 9(2):249–265.

Neal, R. M. (2003). Slice sampling. *The Annals of Statistics*, 31(3):705–741.

Niu, D., Dy, J., and Ghahramani, Z. (2012). A nonparametric bayesian model for multiple clustering with overlapping feature views. *Journal of Machine Learning Research*, 22:814–822.

Pearson, K. (1901). On lines and planes of closest fit to systems of points in space. *Philosophical Magazine Series 6*, 2:559–572.

Pitman, J. (2002). Combinatorial stochastic processes. Technical report, Department of Statistics, University of California at Berkeley.

Rai, P. and Daumé III, H. (2008). The infinite hierarchical factor regression model. In *Advances in Neural Information Processing Systems (NIPS)*.

Rowe, D. B. and Press, S. J. (1998). Gibbs sampling and hill climbing in Bayesian factor analysis. Technical Report 255, Department of Statistics, University of California Riverside.

Roweis, S. (1998). EM algorithms for PCA and SPCA. In *Advances in Neural Information Processing Systems (NIPS)*, pages 626–632. MIT Press.

Sanche, R. and Lonergan, K. (2006). Variable reduction for predictive modeling with clustering. In *Casualty Actuarial Society Forum*, pages 89–100.

Shafto, P., Kemp, C., Mansinghka, V., Gordon, M., and Tenenbaum, J. (2006). Learning cross-cutting systems of categories. In *Proceedings of the 28th annual conference of the Cognitive Science Society*, pages 2146–2151.

Silva, R., Scheines, R., Glymour, C., and Spirtes, P. (2006). Learning the structure of linear latent variable models. *The Journal of Machine Learning Research*, 7:191–246.

Tipping, M. E. and Bishop, C. M. (1999). Probabilistic principal component analysis. *Journal of the Royal Statistical Society. Series B (Statistical Methodology)*, 61(3):611–622.

Vigneau, E. and Qannari, E. (2003). Clustering of variables around latent components. *Communications in Statistics-Simulation and Computation*, 32(4):1131–1150.

West, M., Chang, J., Lucas, J., Nevins, J. R., Wang, Q., and Carvalho, C. (2007). High-dimensional sparse factor modelling: Applications in gene expression genomics. Technical report, ISDS, Duke University.

Winn, J. and Bishop, C. M. (2006). Variational message passing. *Journal of Machine Learning Research*, 6(1):661.

Young, G. (1941). Maximum likelihood estimation and factor analysis. *Psychometrika*, 6(1):49–53.

